# On the Use of Evidence in Neural Networks

**David H. Wolpert**
The Santa Fe Institute
1660 Old Pecos Trail
Santa Fe, NM 87501

## Abstract

The Bayesian "evidence" approximation has recently been employed to determine the noise and weight-penalty terms used in back-propagation. This paper shows that for neural nets it is far easier to use the exact result than it is to use the evidence approximation. Moreover, unlike the evidence approximation, the exact result neither has to be re-calculated for every new data set, nor requires the running of computer code (the exact result is closed form). In addition, it turns out that the evidence procedure's MAP estimate for neural nets is, *in toto*, approximation error. Another advantage of the exact analysis is that it does not lead one to incorrect intuition, like the claim that using evidence one can "evaluate different priors in light of the data". This paper also discusses sufficiency conditions for the evidence approximation to hold, why it can sometimes give "reasonable" results, etc.

## 1   THE EVIDENCE APPROXIMATION

It has recently become popular to consider the problem of training neural nets from a Bayesian viewpoint (Buntine and Weigend 1991, MacKay 1992). The usual way of doing this starts by assuming that there is some underlying target function f from $\mathbf{R}^n$ to $\mathbf{R}$, parameterized by an N-dimensional weight vector $\mathbf{w}$. We are provided with a training set L of noise-corrupted samples of f. Our goal is to make a guess for $\mathbf{w}$, basing that guess only on L. Now assume we have i.i.d. additive gaussian noise resulting in $P(L \mid \mathbf{w}, \beta) \propto \exp(-\beta \chi^2))$, where $\chi^2(\mathbf{w}, L)$ is the usual sum-squared training set error, and $\beta$ reflects the noise level. Assume further that $P(\mathbf{w} \mid \alpha) \propto \exp(-\alpha W(\mathbf{w}))$, where $W(\mathbf{w})$ is the sum of the squares of the weights. If the values of $\alpha$ and $\beta$ are known and fixed, to the values $\alpha_t$ and $\beta_t$ respectively, then $P(\mathbf{w})$

= $P(\mathbf{w} \mid \alpha_t)$ and $P(L \mid \mathbf{w}) = P(L \mid \mathbf{w}, \beta_t)$. Bayes' theorem then tells us that the *posterior* is proportional to the product of the *likelihood* and the *prior*, i.e., $P(\mathbf{w} \mid L) \propto P(L \mid \mathbf{w}) P(\mathbf{w})$. Consequently, finding the *maximum a posteriori* (MAP) $\mathbf{w}$ - the $\mathbf{w}$ which maximizes $P(\mathbf{w} \mid L)$ - is equivalent to finding the $\mathbf{w}$ minimizing $\chi^2(\mathbf{w}, L) + (\alpha_t / \beta_t)W(\mathbf{w})$. This can be viewed as a justification for performing gradient descent with weight-decay.

One of the difficulties with the foregoing is that we almost never know $\alpha_t$ and $\beta_t$ in real-world problems. One way to deal with this is to estimate $\alpha_t$ and $\beta_t$, for example via a technique like cross-validation. In contrast, a Bayesian approach to this problem would be to set priors over $\alpha$ and $\beta$, and then examine the consequences for the posterior of $\mathbf{w}$.

This Bayesian approach is the starting point for the "evidence" approximation created by Gull (Gull 1989). One makes three assumptions, for $P(\mathbf{w} \mid \gamma)$, $P(L \mid \mathbf{w}, \gamma)$, and $P(\gamma)$. (For simplicity of the exposition, from now on the two hyperparameters $\alpha$ and $\beta$ will be expressed as the two components of the single vector $\gamma$.) The quantity of interest is the posterior:

$$P(\mathbf{w} \mid L) = \int d\gamma\, P(\mathbf{w}, \gamma \mid L)$$
$$= \int d\gamma\, [\{P(\mathbf{w}, \gamma \mid L) / P(\gamma \mid L)\} \times P(\gamma \mid L)] \qquad (1)$$

The evidence approximation suggests that if $P(\gamma \mid L)$ is sharply peaked about $\gamma = \gamma'$, while the term in curly brackets is smooth about $\gamma = \gamma'$, then one can approximate the $\mathbf{w}$-dependence of $P(\mathbf{w} \mid L)$ as $P(\mathbf{w}, \gamma' \mid L) / P(\gamma' \mid L) \propto P(L \mid \mathbf{w}, \gamma') P(\mathbf{w} \mid \gamma')$. In other words, with the evidence approximation, one sets the posterior by taking $P(\mathbf{w}) = P(\mathbf{w} \mid \gamma')$ and $P(L \mid \mathbf{w}) = P(L \mid \mathbf{w}, \gamma')$, where $\gamma'$ is the MAP $\gamma$. $P(L \mid \gamma) = \int d\mathbf{w}\, [P(L \mid \mathbf{w}, \gamma) P(\mathbf{w} \mid \gamma)]$ is known as the "evidence" for L given $\gamma$. For relatively smooth $P(\gamma)$, the peak of $P(\gamma \mid L)$ is the same as the peak of the evidence (hence the name "evidence approximation"). Although the current discussion will only explicitly consider using evidence to set hyperparameters like $\alpha$ and $\beta$, most of what will be said also applies to the use of evidence to set other characteristics of the learner, like its architecture.

MacKay has applied the evidence approximation to finding the posterior for the neural net $P(\mathbf{w} \mid \alpha)$ and $P(L \mid \mathbf{w}, \beta)$ recounted above combined with a $P(\gamma) = P(\alpha, \beta)$ which is uniform over all $\alpha$ and $\beta$ from 0 to $+\infty$ (MacKay 1992). In addition to the error introduced by the evidence approximation, additional error is introduced by his need to approximate $\gamma'$. MacKay states that although he expects his approximation for $\gamma'$ to be valid, "it is a matter of further research to establish [conditions for] this approximation to be reliable".

## 2   THE EXACT CALCULATION

It is always true that the *exact* posterior is given by

$$P(\mathbf{w}) = \int d\gamma\, P(\mathbf{w} \mid \gamma)\, P(\gamma),$$
$$P(L \mid \mathbf{w}) = \int d\gamma\, \{P(L \mid \mathbf{w}, \gamma) \times P(\mathbf{w} \mid \gamma) \times P(\gamma)\} / P(\mathbf{w});$$
$$P(\mathbf{w} \mid L) \propto \int d\gamma\, \{P(L \mid \mathbf{w}, \gamma) \times P(\mathbf{w} \mid \gamma) \times P(\gamma)\} \qquad (2)$$

where the proportionality constant, being independent of $\mathbf{w}$, is irrelevant.

Using the neural net $P(\mathbf{w} \mid \alpha)$ and $P(L \mid \mathbf{w}, \beta)$ recounted above, and MacKay's $P(\gamma)$, it is trivial to use equation 2 to calculate that $P(\mathbf{w}) \propto [W(\mathbf{w})]^{-(N/2 + 1)}$, where N is the number of weights. Similarly, with m the number of pairs in L, $P(L \mid \mathbf{w}) \propto [\chi^2(\mathbf{w}, L)]^{-(m/2 + 1)}$. (See (Wolpert 1992) and (Buntine and Weigend 1991), and allow the output values in L to range

from $-\infty$ to $+\infty$.) These two results give us the exact expression for the posterior $P(w \mid L)$. In contrast, the evidence-approximated posterior $\propto \exp[-\alpha'(L) W(w) - \beta'(L) \chi2(w, L)]$.

It is illuminating to compare this exact calculation to the calculation based on the evidence approximation. A good deal of relatively complicated mathematics followed by some computer-based numerical estimation is necessary to arrive at the answer given by the evidence approximation. (This is due to the need to approximate $\gamma$.) In contrast, to perform the exact calculation one only need evaluate a simple gaussian integral, which can be done in closed form, and in particular one doesn't need to perform any computer-based numerical estimation. In addition, with the evidence procedure $\gamma$ must be re-evaluated for each new data set, which means that the formula giving the posterior must be re-derived every time one uses a new data set. In contrast, the exact calculation's formula for the posterior holds for any data set; no re-calculations are required. So as a practical tool, the exact calculation is both far simpler and quicker to use than the calculation based on the evidence approximation.

Another advantage of the exact calculation, of course, is that it is *exact*. Indeed, consider the simple case where the noise is fixed, i.e., $P(\gamma) = P(\gamma_1) \delta(\gamma_2 - \beta_t)$, so that the only term we must "deal with" is $\gamma_1 = \alpha$. Set all other distributions as in (MacKay 1992). For this case, the w-dependence of the exact posterior can be quite different from that of the evidence-approximated posterior. In particular, note that the MAP estimate based on the exact calculation is $w = 0$. This is, of course, a silly answer, and reflects the poor choice of distributions made in (MacKay 1992). In particular, it directly reflects the un-normalizability of MacKay's $P(\alpha)$. However the important point is that this is the *exactly correct* answer for those distributions. On the other hand, the evidence procedure will result in an MAP estimate of $\text{argmin}_w [\chi^2(w, L) + (\alpha' / \beta')W(w)]$, where $\alpha'$ and $\beta'$ are derived from L. This answer will often be far from the correct answer of $w = 0$. Note also that the evidence approximations's answer will vary, perhaps greatly, with L, whereas the correct answer is L-independent. Finally, since the correct answer is $w = 0$, the difference between the evidence procedure's answer and the correct answer is equal to the evidence procedure's answer. In other words, although there exist scenarios for which the evidence approximation is valid, neural nets with flat $P(\gamma_1)$ is not one of them; for this scenario, the evidence procedure's answer is *in toto* approximation error. (A possible reason for this is presented in section 4.)

If one were to use a more reasonable $P(\alpha)$, uniform only from 0 to an upper cut-off $\alpha_{max}$, the results would be essentially the same, for large enough $\alpha_{max}$. The effect on the exact posterior, to first order, is to introduce a small region around $w = 0$ in which $P(w)$ behaves like a decaying exponential in $W(w)$ (the exponent being set by $\alpha_{max}$) rather than like $[W(w)]^{-(N/2 + 1)}$ (T. Wallstrom, private communication). For large enough $\alpha_{max}$, the region is small enough so that the exact posterior still has a peak very close to $0$. On the other hand, for large enough $\alpha_{max}$, there is no change in the evidence procedure's answer. (Generically, the major effect on the evidence procedure of modifying $P(\gamma)$ is not to change its guess for $P(w \mid L)$, but rather to change the associated error, i.e., change whether sufficiency conditions for the validity of the approximation are met. See below.) Even with a normalizable prior, the evidence procedure's answer is still essentially all approximation error.

Consider again the case where the prior over both $\alpha$ and $\beta$ is uniform. With the evidence approximation, the log of the posterior is $-\{ \chi^2(w, L) + (\alpha' / \beta')W(w) \}$, where $\alpha'$ and $\beta'$ are set by the data. On the other hand, the exact calculation shows that the log of the pos-

terior is really given by $-\{\ln[\chi^2(w, L)] + (N+2/m+2)\ln[W(w)]\}$. What's interesting about this is not simply the logarithms, absent from the evidence approximation's answer, but also the factor multiplying the term involving the "weight penalty" quantity $W(w)$. In the evidence approximation, this factor is data-dependent, whereas in the exact calculation it only depends on the number of data. Moreover, the value of this factor in the exact calculation tells us that if the number of weights increases, or alternatively the number of training examples decreases, the "weight penalty" term becomes more important, and fitting the training examples becomes less important. (It is not at all clear that this trade-off between N and m is reflected in $(\alpha' / \beta')$, the corresponding factor from the evidence approximation.) As before, if we have upper cut-offs on $P(\gamma)$, so that the MAP estimate may be reasonable, things don't change much. For such a scenario, the N vs. m trade-off governing the relative importance of $W(w)$ and $\chi^2(w, L)$ still holds, but only to lowest order, and only in the region sufficiently far from the ex-singularities (like $w = 0$) so that $P(w \mid L)$ behaves like $[W(w)]^{-(N/2 + 1)} \times [\chi^2(w, L)]^{-(m/2 + 1)}$.

All of this notwithstanding, the evidence approximation has been reported to give good results in practice. This should not be all that surprising. There are many procedures which are formally illegal but which still give reasonable advice. (Some might classify all of non-Bayesian statistics that way.) The evidence procedure fixes $\gamma$ to a single value, essentially by maximum likelihood. That's not unreasonable, just usually illegal (as well as far more laborious than the correct Bayesian procedure).

In addition, the tests of the evidence approximation reported in (MacKay 1992) are not all that convincing. For paper 1, the evidence approximation gives $\alpha' = 2.5$. For any other $\alpha$ in an interval extending *three orders of magnitude* about this $\alpha'$, test set error is essentially unchanged (see figure 5 of (MacKay 1992)). Since such error is what we're ultimately interested in, this is hardly a difficult test of the evidence approximation. In paper 2 of (MacKay 1992) the initial use of the evidence approximation is "a failure of Bayesian prediction"; $P(\gamma \mid L)$ doesn't correlate with test set error (see figure 7). MacKay addresses this by arguing that poor Bayesian results are never wrong, but only "an opportunity to learn" (in contrast to poor non-Bayesian results?). Accordingly, he modifies the system *while looking at the test set*, to get his desired correlation on the test set. To do this legally, he should have instead modified his system while looking at a validation set, separate from the test set. However if he had done that, it would have raised the question of why one should use evidence at all; since one is already assuming that behavior on a validation set corresponds to behavior on a test set, why not just set $\alpha$ and $\beta$ via cross-validation?

## 3   EVIDENCE AND THE PRIOR

Consider again equation 1. Since $\gamma'$ depends on the data L, it would appear that when the evidence approximation is valid, the data determines the prior, or as MacKay puts it, "the modern Bayesian ... does not assign the priors - many different priors can be ... compared in the light of the data by evaluating the evidence" (MacKay 1992). If this were true, it would remove perhaps the most major objection which has been raised concerning Bayesian analysis - the need to choose priors in a subjective manner, independent of the data. However the exact $P(w)$ given by equation 2 is data-independent. So one *has* chosen the prior, in a subjective way. The evidence procedure is simply providing a data-dependent approximation to a data-independent quantity. In no sense does the evidence procedure allow one to side-step the need to make subjective assumptions which fix $P(w)$.

Since the true P(w) doesn't vary with L whereas the evidence approximation's P(w) does, one might suspect that that approximation to P(w) can be quite poor, even when the evidence approximation to the posterior is good. Indeed, if $P(w \mid \gamma_1)$ is exponential, there is no non-pathological scenario for which the evidence approximation to P(w) is correct:

**Theorem 1:** *Assume that* $P(w \mid \gamma_1) \propto e^{-\gamma_1 U(w)}$. *Then the only way that one can have* $P(w) \propto e^{-\alpha U(w)}$ *for some constant* $\alpha$ *is if* $P(\gamma_1) = 0$ *for all* $\gamma_1 \neq \alpha$.

**Proof:** Our proposed equality is $\exp(-\alpha \times U) = \int d\gamma_1 \{ P(\gamma_1) \times \exp(-\gamma_1 \times U) \}$ (the normalization factors having all been absorbed into $P(\gamma_1)$). We must find an $\alpha$ and a normalizable $P(\gamma_1)$ such that this equality holds for all allowed U. Let u be such an allowed value of U. Take the derivative with respect to U of both sides of the proposed equality t times, and evaluate for U = u. The result is $\alpha^t = \int d\gamma_1 ((\gamma_1)^t \times R(\gamma_1))$ for any integer $t \geq 0$, where $R(\gamma_1) \equiv$ $P(\gamma_1) \exp(u(\alpha - \gamma_1))$. Using this, we see that $\int d\gamma_1 ((\gamma_1 - \alpha)^2 \times R(\gamma_1)) = 0$. Since both $R(\gamma_1)$ and $(\gamma_1 - \alpha)^2$ are nowhere negative, this means that for all $\gamma_1$ for which $(\gamma_1 - \alpha)^2 \neq 0$, $R(\gamma_1)$ must equal zero. Therefore $R(\gamma_1)$ must equal zero for all $\gamma_1 \neq \alpha$. QED.

Since the evidence approximation for the prior is always wrong, how can its approximation for the posterior ever be good? To answer this, write $P(w \mid L) = P(L \mid w) \times [P'(w) + E(w)] / P(L)$, where P'(w) is the evidence approximation to P(w). (It is assumed that we know the likelihood exactly.) This means that $P(w \mid L) - \{ P(L \mid w) \times P'(w) / P(L) \}$, the error in the evidence procedure's estimate for the posterior, equals $P(L \mid w) \times E(w) / P(L)$. So we *can* have arbitrarily large E(w) and not introduce sizable error into the posterior of w, but only for those w for which $P(L \mid w)$ is small. As L varies, the w with non-negligible likelihood vary, and the $\gamma$ such that *for those* w $P(w \mid \gamma)$ is a good approximation to P(w) varies. When it works, the $\gamma'$ given by the evidence approximation reflects this changing of $\gamma$ with L.

# 4   SUFFICIENCY CONDITIONS FOR EVIDENCE TO WORK

Note that regardless of how peaked the evidence is, $-\{ \chi^2(w, L) + (\alpha' / \beta')W(w) \} \neq$ $-\{ \ln[\chi^2(w, L)] + (N+2 / m+2) \ln[W(w)] \}$; the evidence approximation always has non-negligible error for neural nets used with flat $P(\gamma)$. To understand this, one must carefully elucidate a set of sufficiency conditions necessary for the evidence approximation to be valid. (Unfortunately, this has never been done before. A direct consequence is that no one has ever checked, formally, that a particular use of the evidence approximation is justified.)

One such set of sufficiency conditions, the one implicit in all attempts to date to justify the evidence approximation (i.e., the one implicit in the logic of equation 1), is the following:

$P(\gamma \mid L)$ is sharply peaked about a particular $\gamma$, $\gamma'$.    (i)
$P(w, \gamma \mid L) / P(\gamma \mid L)$ varies slowly around $\gamma = \gamma'$.    (ii)
$P(w, \gamma \mid L)$ is infinitesimal for all $\gamma$ sufficiently far from $\gamma'$.    (iii)

Formally, condition (iii) can be taken to mean that there exists a not too large positive constant k, and a small positive constant $\delta$, such that $\mid P(w \mid L) - k \int_{\gamma'-\delta}^{\gamma'+\delta} d\gamma \, P(w, \gamma \mid L) \mid$ is bounded by a small constant $\varepsilon$ for all w. (As stated, (iii) has k = 1. This will almost always

be the case in practice and will usually be assumed, but it is not needed to prove theorem 2.) Condition (ii) can be taken to mean that across $[\gamma - \delta, \gamma + \delta]$, $|P(w \mid \gamma, L) - P(w \mid \gamma' L)| < \tau$, for some small positive constant $\tau$, for all $w$. (Here and throughout this paper, when $\gamma$ is multi-dimensional, "$\delta$" is taken to be a small positive vector.)

**Theorem 2:** *When conditions (i), (ii), and (iii) hold,* $P(w \mid L) \cong P(L \mid w, \gamma) \times P(w \mid \gamma)$, *up to an (irrelevant) overall proportionality constant.*

**Proof:** Condition (iii) gives $| P(w \mid L) - k \int_{\gamma-\delta}^{\gamma+\delta} d\gamma \, [P(w \mid \gamma L) \times P(\gamma \mid L)] | < \varepsilon$ for all $w$. However $| k \int_{\gamma-\delta}^{\gamma+\delta} d\gamma \, [P(w \mid \gamma, L) \times P(\gamma \mid L)] - k \, P(w \mid \gamma' L) \int_{\gamma-\delta}^{\gamma+\delta} d\gamma \, P(\gamma \mid L) | < \tau k \times \int_{\gamma-\delta}^{\gamma+\delta} d\gamma \, P(\gamma \mid L)$, by condition (ii). If we now combine these two results, we see that $| P(w \mid L) - k \, P(w \mid \gamma' L) \int_{\gamma-\delta}^{\gamma+\delta} d\gamma \, P(\gamma \mid L) | < \varepsilon + \tau k \times \int_{\gamma-\delta}^{\gamma+\delta} d\gamma \, P(\gamma \mid L)$. Since the integral is bounded by 1, $| P(w \mid L) - k \, P(w \mid \gamma' L) \int_{\gamma-\delta}^{\gamma+\delta} d\gamma \, P(\gamma \mid L) | < \varepsilon + \tau k$. Since the integral is independent of $w$, up to an overall proportionality constant (that integral times $k$) the $w$-dependence of $P(w \mid L)$ can be approximated by that of $P(w \mid \gamma', L) \propto P(L \mid w, \gamma') \times P(w \mid \gamma')$, incurring error less than $\varepsilon + \tau k$. Take $k$ not too large and both $\varepsilon$ and $\tau$ small. QED.

Note that the proof would go through even if $P(\gamma \mid L)$ were not peaked about $\gamma'$, or if $P(\gamma \mid L)$ were peaked about some point far from the $\gamma'$ for which (ii) and (iii) hold; nowhere in the proof is the definition of $\gamma'$ from condition (i) used. However in practice, when condition (iii) is met, $k = 1$, $P(\gamma \mid L)$ falls to 0 outside of $[\gamma - \delta, \gamma + \delta]$, and $P(w \mid \gamma, L)$ stays reasonably bounded for all such $\gamma$. (If this weren't the case, then $P(w \mid \gamma, L)$ would have to fall to 0 outside of $[\gamma - \delta, \gamma + \delta]$, something which is rarely true.) So we see that we could either just give conditions (ii) and (iii), or we could give (i), (ii), and the extra condition that $P(w \mid \gamma, L)$ is bounded small enough so that condition (iii) is met. (In addition, one can prove that if the evidence approximation is valid, then conditions (i) and (ii) give condition (iii).)

In any case, it should be noted that conditions (i) and (ii) by themselves are *not* sufficient for the evidence approximation to be valid. To see this, have $w$ be one-dimensional, and let $P(w, \gamma \mid L) = 0$ both for $\{|\gamma - \gamma'| < \delta, |w - w^*| < v\}$ and for $\{|\gamma - \gamma'| > \delta, |w - w^*| > v\}$. Let it be constant everywhere else (within certain bounds of allowed $\gamma$ and $w$). Then for both $\delta$ and $v$ small, conditions (i) and (ii) hold: the evidence is peaked about $\gamma'$, and $\tau = 0$. Yet for the true MAP $w$, $w^*$, the evidence approximation fails badly. (Generically, this scenario will also result in a big error if rather than using the evidence-approximated posterior to guess the MAP $w$, one instead uses it to evaluate the posterior-averaged $f$, $\int df \, f \, P(f \mid L)$.)

Gull mentions only condition (i). MacKay also mentions condition (ii), but not condition (iii). Neither author plugs in for $\varepsilon$ and $\tau$, or in any other way uses their distributions to infer bounds on the error accompanying their use of the evidence approximation.

Since by (i) $P(\gamma \mid L)$ is sharply peaked about $\gamma'$, one would expect that for (ii) to hold $P(w, \gamma \mid L)$ must also be sharply peaked about $\gamma'$. Although this line of reasoning can be formalized, it turns out to be easier to prove the result using sufficiency condition (iii):

**Theorem 3:** *If condition (iii) holds, then for all $w$ such that $P(w \mid L) > c > \varepsilon$, for each component $i$ of $\gamma$, $P(w, \gamma_i \mid L)$ must have a $\gamma_i$-peak somewhere within $\delta_i[1 + 2\varepsilon / (c - \varepsilon)]$ of $(\gamma')_i$.*

**Proof:** Condition (iii) with $k = 1$ tells us that $P(w \mid L) - \int_{\gamma-\delta}^{\gamma+\delta} \delta\gamma \, P(w, \gamma \mid L) < \varepsilon$. Extending

the integrals over $\gamma_{j \neq i}$ gives $P(w \mid L) - \int_{(\gamma-\delta)_i}^{(\gamma+\delta)_i} d\gamma_i \, P(w, \gamma_i \mid L) < \varepsilon$. From now on the i

subscript on $\gamma$ and $\delta$ will be implicit. We have $\varepsilon > \int_{\gamma+\delta}^{\gamma+\delta+r} d\gamma \, P(w, \gamma \mid L)$ for any scalar r

$> 0$. Assume that $P(w, \gamma \mid L)$ doesn't have a peak anywhere in $[\gamma - \delta, \gamma + \delta + r]$. Without loss of generality, assume also that $P(w, \gamma + \delta \mid L) \geq P(w, \gamma - \delta \mid L)$. These two assumptions mean that for any $\gamma \in [\gamma + \delta, \gamma + \delta + r]$, the value of $P(w, \gamma \mid L)$ exceeds the maximal value

it takes on in the interval $[\gamma - \delta, \gamma + \delta]$. Therefore $\int_{\gamma+\delta}^{\gamma+\delta+r} d\gamma \, P(w, \gamma \mid L) \geq (r / 2\delta) \times$

$\int_{\gamma-\delta}^{\gamma+\delta} d\gamma \, P(w, \gamma \mid L)$. This means that $\int_{\gamma-\delta}^{\gamma+\delta} d\gamma \, P(w, \gamma \mid L) < 2\delta\varepsilon / r$. But since $P(w \mid L) <$

$\varepsilon + \int_{\gamma-\delta}^{\gamma+\delta} d\gamma \, P(w, \gamma \mid L)$, this means that $P(w \mid L) < \varepsilon(1 + 2\delta / r)$. So if $P(w \mid L) > c > \varepsilon$, r $< 2\varepsilon / (c - \varepsilon)$, and there must be a peak of $P(w, \gamma \mid L)$ within $\delta(1 + 2\varepsilon/(c - \varepsilon))$ of $\gamma$. QED.

So for those w with non-negligible posterior, for $\varepsilon$ small, the $\gamma$-peak of $P(w, \gamma \mid L) \propto P(L \mid w, \gamma) \times P(w \mid \gamma) \times P(\gamma)$ must lie essentially within the peak of $P(\gamma \mid L)$. Therefore:

**Theorem 4:** *Assume that* $P(w \mid \gamma_1) = \exp(-\gamma_1 U(w)) / Z_1(\gamma_1)$ *for some function* $U(.)$, $P(L \mid w, \gamma_2) = \exp(-\gamma_2 V(w, L)) / Z_2(\gamma_2, w)$ *for some function* $V(., .)$, *and* $P(\gamma) = P(\gamma_1)P(\gamma_2)$. *(The* $Z_i$ *act as normalization constants.) Then if condition (iii) holds, for all w with non-negligible posterior the* $\gamma$-*solution to the equations*

$$-U(w) + \partial_{\gamma_1} [\ln(P(\gamma_1) - \ln(Z_1(\gamma_1))] = 0$$

$$-V(w, L) + \partial_{\gamma_2} [\ln(P(\gamma_2) - \ln(Z_2(\gamma_2, w))] = 0$$

*must like within the* $\gamma$-*peak of* $P(\gamma \mid L)$.

**Proof:** $P(w, \gamma \mid L) \propto \{P(\gamma_1) \times P(\gamma_2) \times \exp[-\gamma_1 U(w) - \gamma_2 V(w, L)] \} / \{Z_1(\gamma_1) \times Z_2(\gamma_2, w)\}$. For both i = 1 and i = 2, evaluate $\partial_{\gamma_i} \{\int d\gamma_{j \neq i} \, P(w, \gamma \mid L)\}$, and set it equal to zero. This gives the two equations. Now define "the $\gamma$-peak of $P(\gamma \mid L)$" to mean a cube with i-component width $\delta_i[1 + 2\varepsilon / (c - \varepsilon)]$, centered on $\gamma$, where having a "non-negligible posterior" means $P(w \mid L) > c$. Applying theorem 3, we get the result claimed. QED.

In particular, in MacKay's scenario, $P(\gamma)$ is uniform, $W(w) = \Sigma_{i=1}^m (w_i)^2$, and $V(w, L) = \chi^2(w, L)$. Therefore $Z_1$ and $Z_2$ are proportional to $(\gamma_1)^{-N/2}$ and $(\gamma_2)^{-m/2}$ respectively. This means that if the vector $\{\gamma_1, \gamma_2\} = \{N / [2W(w)], m / [2\chi^2(w, L)]\}$ does not lie within the peak of the evidence for the MAP w, condition (iii) does not hold. That $\gamma_1 / \gamma_2$ must approximately equal $[N \chi^2(w, L)] / [m W(w)]$ should not be too surprising. If we set the w-gradient of both the evidence-approximated and exact $P(w \mid L)$ to zero, and demand that the same w,w', solves both equations, we get $\gamma_1 / \gamma_2 = -[(N + 2) \chi^2(w', L)] / [(m + 2)W(w')]$. (Unfortunately, if one continues and evaluates $\partial_{w_i}\partial_{w_j}P(w \mid L)$ at w', often one finds that it has opposite signs for the two posteriors - a graphic failure of the evidence approximation.)

It is not clear from the provided neural net data whether this condition is met in (MacKay 1992). However it appears that the corresponding condition is not met, for $\gamma_1$ at least, for the scenario in (Gull 1992) in which the evidence approximation is used with $U(.)$ being the entropy. (See (Strauss et al. 1993, Wolpert et al. 1993).) Since conditions (i) through (iii)

are sufficient conditions, not necessary ones, this does not prove that Gull's use of evidence is invalid. (It is still an open problem to delineate the full iff for when the evidence approximation is valid, though it appears that matching of peaks as in theorem 3 is necessary. See (Wolpert et al. 1993).) However this does mean that the *justification* offered by Gull for his use of evidence is apparently invalid. It might also help explain why Gull's results were "visually disappointing and ... clearly ... 'over-fitted'", to use his terms.

The first equation in theorem 4 can be used to set restrictions on the set of $w$ which both have non-negligible posterior and for which condition (iii) holds. Consider for example MacKay's scenario, where that equation says that $N / 2W(w)$ must lie within the width of the evidence peak. If the evidence peak is sharp, this means that unless all $w$ with non-negligible posterior have essentially the same $W(w)$, condition (iii) can not hold for all of them.

Finally, if for some reason one wishes to know $\gamma'$, theorem 4 can sometimes be used to circumvent the common difficulty of evaluating $P(\gamma | L)$. To do this, one assumes that conditions (i) through (iii) hold. Then one finds *any* $w$ with a non-negligible posterior (say by use of the evidence approximation coupled with approximations to $P(\gamma | L)$) and uses it in theorem 4 to find a $\gamma$ which must lie within the peak of $P(\gamma | L)$, and therefore must lie close to the correct value of $\gamma'$.

To summarize, there might be scenarios in which the exact calculation of the quantity of interest is intractable, so that some approximation like evidence is necessary. Alternatively, if one's choice of $P(w | \gamma)$, $P(\gamma)$, and $P(L | w, \gamma)$ is poor, the evidence approximation would be useful if the error in that approximation somehow "cancels" error in the choice of distributions. However if one believes one's choice of distributions, and if the quantity of interest is $P(w | L)$, then at a minimum one should check conditions (i) through (iii) before using the evidence approximation. When one is dealing with neural nets, one needn't even do that; the exact calculation is quicker and simpler than using the evidence approximation.

## Acknowledgments

This work was done at the SFI and was supported in part by NLM grant F37 LM00011. I would like to thank Charlie Strauss and Tim Wallstrom for stimulating discussion.

## References

Buntine, W., Weigend, A. (1991). Bayesian back-propagation. *Complex Systems*, **5**, 603.

Gull, S.F. (1989). Developments in maximum entropy data analysis. In "Maximum-entropy and Bayesian methods", J. Skilling (Ed.). Kluwer Academics publishers.

MacKay, D.J.C. (1992). Bayesian Interpolation. A Practical Framework for Backpropagation Networks. *Neural Computation*, **4**, 415 and 448.

Strauss, C.E.M, Wolpert, D.H., Wolf, D.R. (1993). Alpha, Evidence, and the Entropic Prior. In "Maximum-entropy and Bayesian methods", A. Mohammed-Djafari (Ed.). Kluwer Academics publishers. In press

Wolpert, D.H. (1992). A Rigorous Investigation of "Evidence" and "Occam Factors" in Bayesian Reasoning. SFI TR 92-03-13. Submitted.

Wolpert, D.H., Strauss, C.E.M., Wolf, D.R. (1993). On evidence and the marginalization of alpha in the entropic prior. In preparation.


